# A Network Mechanism for the Determination of Shape-From-Texture

**Ko Sakai and Leif H. Finkel**
Department of Bioengineering and
Institute of Neurological Sciences
University of Pennsylvania
220 South 33rd Street, Philadelphia, PA 19104-6392
ko@ganymede.seas.upenn.edu, leif@ganymede.seas.upenn.edu

## Abstract

We propose a computational model for how the cortex discriminates shape and depth from texture. The model consists of four stages: (1) extraction of local spatial frequency, (2) frequency characterization, (3) detection of texture compression by normalization, and (4) integration of the normalized frequency over space. The model accounts for a number of psychophysical observations including experiments based on novel random textures. These textures are generated from white noise and manipulated in Fourier domain in order to produce specific frequency spectra. Simulations with a range of stimuli, including real images, show qualitative and quantitative agreement with human perception.

## 1  INTRODUCTION

There are several physical cues to shape and depth which arise from changes in projection as a surface curves away from view, or recedes in perspective. One major cue is the orderly change in the spatial frequency distribution of texture along the surface. In machine vision approaches, various techniques such as Fourier transformation or wavelet decomposition are used to determine spatial frequency spectra across a surface. The determination of the transformation relating these spectra is a difficult problem, and several techniques have been proposed such as an affine transformation (Super and Bovik

1992) or a momentum method (Krumm and Shafer 1992). We address the question of how a biological system which has access only to limited spatial frequency information and has constrained computational capabilities can nonetheless accurately determine shape and depth from texture. For example, the visual system might avoid the direct comparison of frequency spectra themselves and instead rely on a simpler characterization of the spectra such as the mean frequency, peak frequency, or the gradient of a frequency component (Sakai and Finkel 1993; Turner, Gerstein, Bajcsy 1991). In order to study what frequency information is actually utilized by humans, we created novel random texture patterns and carried out psychophysical experiments with these stimuli. These patterns are generated by manipulating the frequency components of white noise stimuli in the Fourier domain so as to produce stimuli with exactly specified frequency spectra. Based on these experiments, we propose a network mechanism for the perception of shape-from-texture which takes into account physiological and anatomical constraints as well as computational considerations.

## 2  MODEL FOR SHAPE FROM TEXTURE

The model consists of four major processes: extraction of the local spatial frequency at each orientation, frequency characterization, determination of texture compression by frequency normalization, and the integration of the normalized frequency over space. A schematic illustration of the model is shown in figure 1. Our psychophysical experiments suggest that the visual system may use *spatially averaged peak frequency* for characterizing the frequency distribution. The change of surface orientation is determined from the locally aligned compression of texture which is detected by frequency normalization followed by lateral inhibition among different orientations. Depth is then computed from the integration of the normalized frequency over space. The model is implemented in feed-forward distributed networks and simulated using the NEXUS neural network simulator (Sajda, Sakai, Yen and Finkel 1993).

## 3  MOTIVATION FOR EACH STAGE OF THE MODEL

The frequency extraction is carried out by units modeling complex cells in area V1. These units have subunits with On and Off center difference of Gaussian(DOG) masks tuned to specific frequencies and orientations. The units take local maximum of the subunits. As in energy-based models (Bergen and Adelson 1989; Malik and Perona 1990), these units accomplish some major aspects of complex cell functions in the space domain including invariance to the direction of contrast and spatial phase.

The second stage of the model extracts spatially averaged peak frequency. In order to examine what frequency information is actually utilized by humans, we created random texture patterns with specific frequency spectra generated by manipulating the frequency components of a white noise pattern in Fourier domain. Figure 2 shows a vertical cylinder and a tilted perspective plane constructed by this technique from white noise. We are able to see the three dimensional shape of the cylinder in (1). The stimuli were constructed by making each frequency component undergo a step change at some

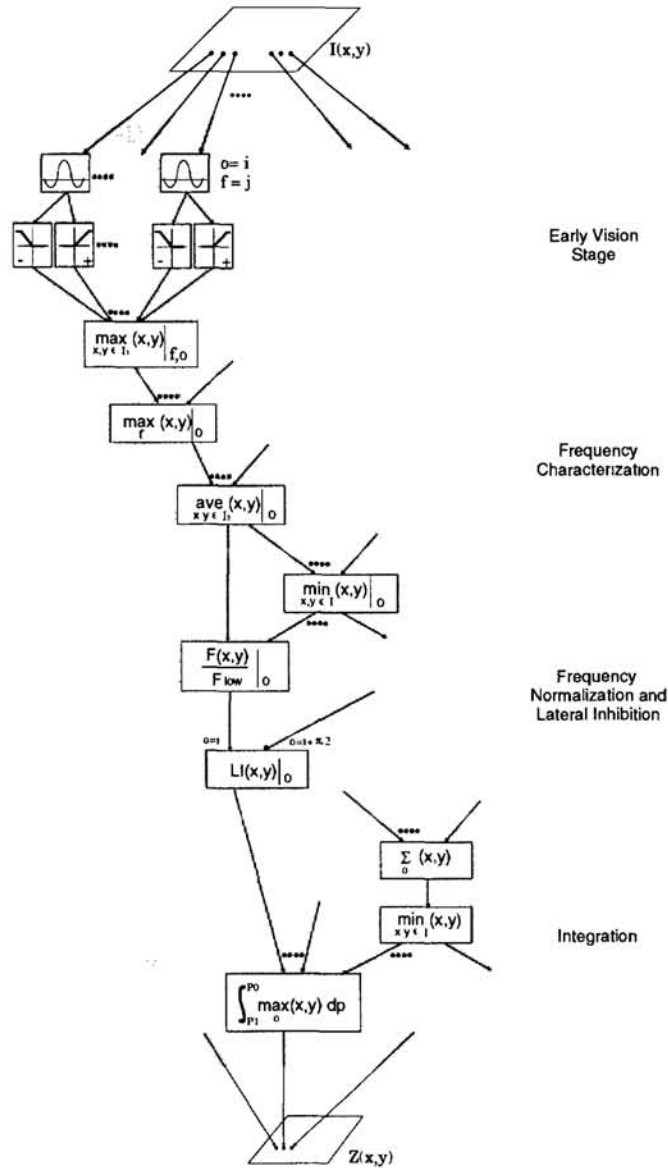

Figure 1. A schematic illustration of the shape-from-texture model consisting of four major stages. The early vision stage models major spatial properties of complex cells in order to decompose local spatial frequency. The second stage characterizes the frequency by the spatially averaged peak frequency. The third stage detects locally aligned texture compression by normalizing frequency and taking lateral inhibition among orientation channels. The last stage determines 3D depth by integrating the amount of texture compression - which corresponds to the local surface slant. Indices "f" and "o" denote frequency and orientation channels, respectively. max, min, ave, and LI stand for taking maximum, minimum, average, and lateral inhibition. The vertical bar indicates that the function is processed independently within each of denoted channels.

position along the cylinder; higher frequencies undergo the change at positions closer to the cylinder's edges. Since the gradient of each frequency component is always either zero or infinity, this suggests that gradients of individual frequency components over space do not serve as a dominant cue for three dimensional shape perception. Similar experiments have been conducted using various stimuli with controlled frequency spectra. The results of these experiments suggest that averaged peak frequency is a strong cue for the human perception of three dimensional shape and depth.

The third stage of the model normalizes local frequencies by the global lowest frequency on the surface. We assume that the region containing the global lowest frequency is the frontal plane standing vertically with respect to the viewer. One of the justifications for this assumption can be seen in simple artificial images shown in figure 3. In both (1) and (2), the bottom region looks vertical to us, and the planes above this region looks slanted, although the patterns of the center region of (1) and the lower region of (2) are identical.

From a computational point of view, the normalization of frequency corresponds to an approximation of the relation between local slant and spatial frequency. Depth, Z, as a function of X (see figure 4) is given by:

$$Z(x) = \int_{X_0}^{X} \tan\{ \cos^{-1}(\frac{Fo}{F(x)}) \}dx = \int_{X_0}^{X} \sqrt{(\frac{F(x)}{Fo})^2 - 1}\ dx \qquad \text{eq.(1)}$$

where Fo is the global lowest frequency. Considering a boundary condition, $Z(x) = 0$, if $F(x) = Fo$, the integrand can be reasonably approximated by $(F(x) - F_0) / F_0$. The second stage of the model actually computes this value, and a later stage carries out the integration.

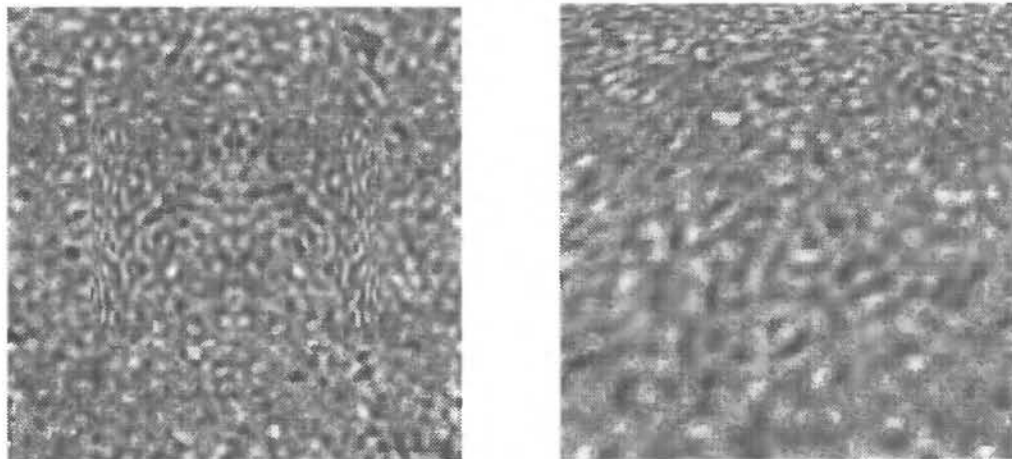

Figure 2. Random texture patterns generated by manipulating the frequency components of white noise in Fourier domain. A horizontal cylinder embedded in white noise (1), and a tilted plane (2).

The second half of this stage detects the local alignment of texture compression. This local alignment is detected by taking the lateral inhibition of normalized frequencies among different orientations. Recent psychophysical experiments (Todd and Akerstrom 1987; Cumming, Johnson, and Parker 1993) show that the compression of texture in a single orientation is a cue for the perception of shape-from-texture. We can confirm this result from figure 5. Three images on the top of this figure have compression in a single orientation, but those on the bottom do not. We clearly see smooth three dimensional ellipsoids from the top images but not from the bottom images.

The last stage of the model computes the integral of the normalized frequency in order to obtain depth. This integration begins from the region with lowest spatial frequency and follows the path of the local steepest descent in spatial frequency.

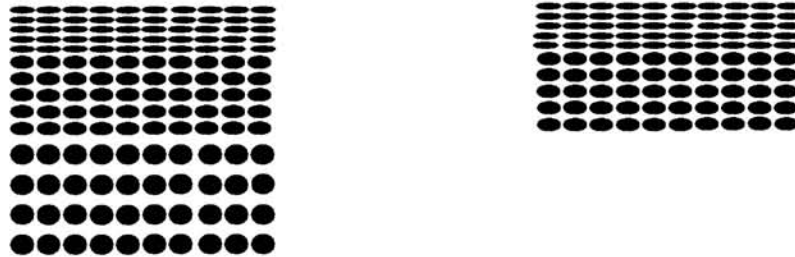

Figure 3. Objects consist of three planes(left), and two planes(right). In both stimuli, the bottom regions look vertical to us, and the planes above this region look slanted, although the patterns of the center region of (1) and the lower region of (2) are identical.

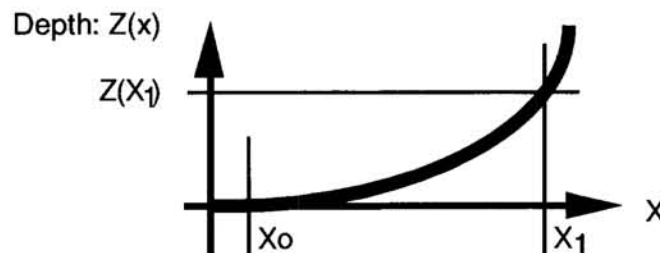

Figure 4. The coordinate system for the equation (1). Depth, Z, is given as a function of position, X.

## 4   SIMULATIONS

A quantitative test of the model was carried out by constructing ellipsoids with different eccentricities and texture patterns shown in figure 5. Results are plotted in figure 6. For the regular ellipsoids, there is a linear relation between real depth and that determined by the model. This linear relation agrees with psychophysical experiments (Todd and Akerstrom 1987; Bülthoff 1991) showing similar human performance for such stimuli. All of the irregular texture patterns produced little perception of depth, in agreement with human performance.

Many artificial and real images have been tested with the model and show good agreement with human perception. For an example, a real image of a part of cantaloupe, and its computed depth are shown in figure 7. Real images were obtained with a CCD camera and were input to NEXUS via an Imaging Technology's S151 image processor.

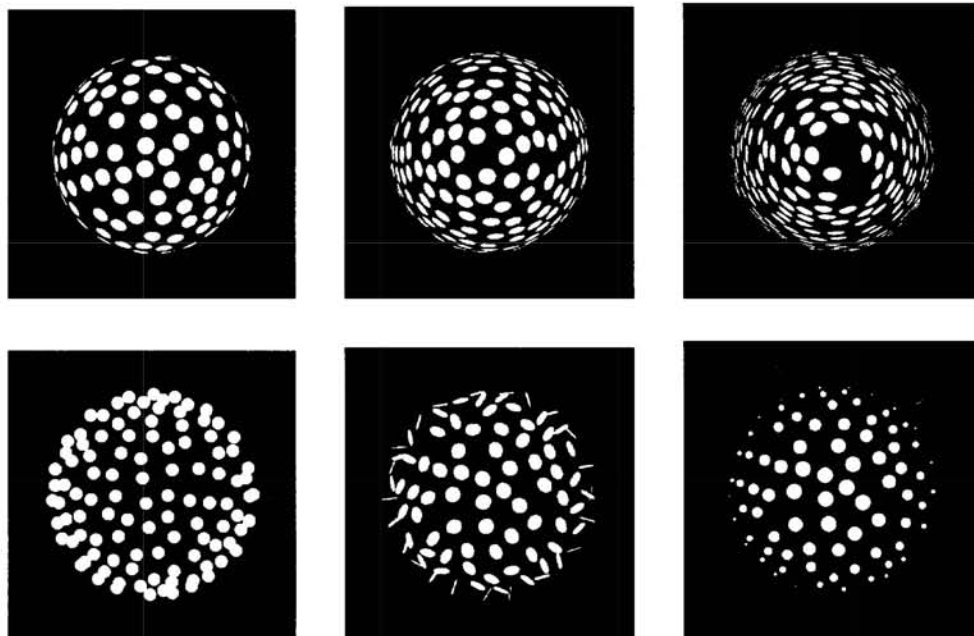

Figure 5. (Top) Regular ellipsoids with eccentricities of 1,2, and 4. (Bottom) Irregular texture patterns: (left) no compression with regular density change, (middle) randomly oriented regular compression, (right) pan-orientational regular compression.

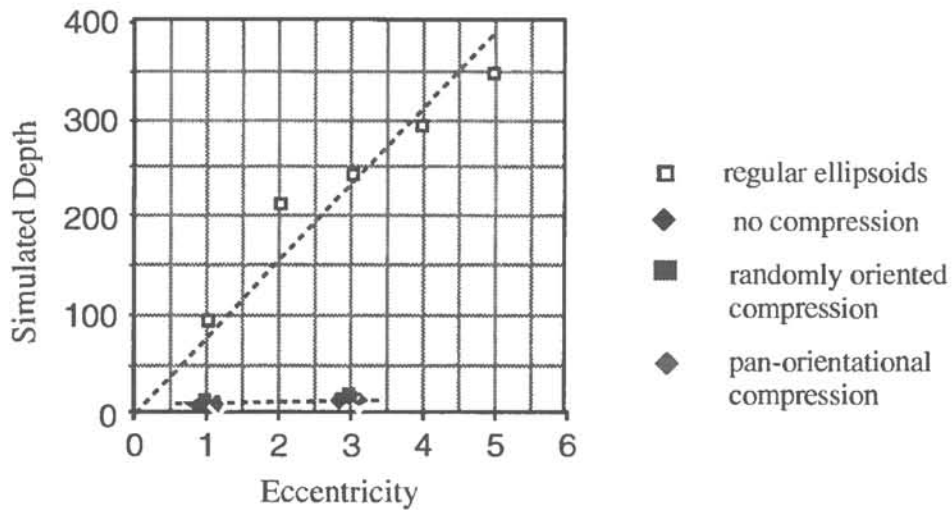

Figure 6. Depth perceived by the model as a function of actual eccentricity. The simulated depth of regular ellipsoids shows a linear relation to the actual depth. Irregular patterns produced little depth, in agreement with human perception.

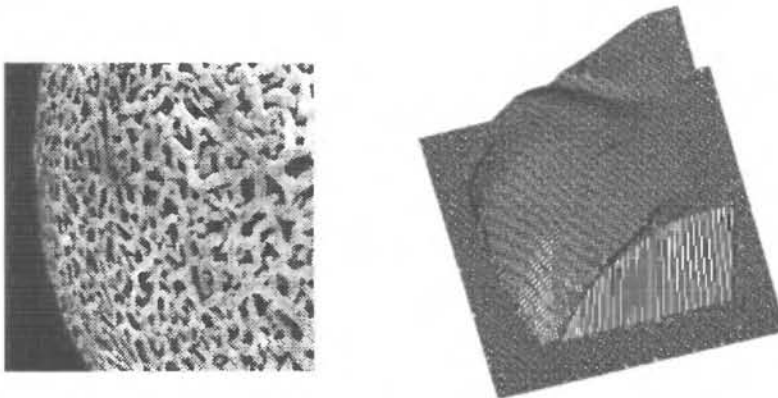

Figure 7. An example of the model's response to a real image. A part of cantaloupe (left), and its depth computed by the model(right).

## 5 CONCLUSIONS

(1) We propose a biologically-based network model of shape-from-texture based on the determination of change in spatial frequency.

(2) Preliminary psychophysical evidence suggests that the spatially averaged peak frequency is employed to characterize the spatial frequency distribution rather than using a frequency spectrum or each component of frequency.

(3) This characterization is validated by psychophysical experiments using novel random textures with specified frequency spectra. The patterns are generated from white noise and manipulated in Fourier domain in order to realize specific frequency characteristics.

(4) The model has been tested with a number of artificial stimuli and real images taken by video camera. Responses show qualitative and quantitative agreements with human perception.

## Acknowledgments

This work is supported by grants from The Office of Naval Research (N00014-90-J-1864, N00014-93-1-0681), The Whitaker Foundation, and The McDonnell-Pew Program in Cognitive Neuroscience.

## References

Super, B.J. and Bovik, A.C. (1992), Shape-from-texture by wavelet-based measurement of local spectral moments. *Proc. IEEE CVPR 1992*, p296-300

Krumm, J. and Shafer, S.A. (1992), Shape from periodic texture using the spectrogram. *Proc. IEEE CVPR 1992*, p284-289

Sakai, K. and Finkel, L.H. (1994), A cortical mechanism underlying the perception of shape-from-texture. In F.Eeckman, et al.(ed.), *Computation and Neural Systems 1993*, Norwell, MA: Kluwer Academic Publisher [in press]

Sajda, P., Sakai, K., Yen, S-C., and Finkel, L.H. (1993), In Skrzypek, J. (ed.), *Neural Network Simulation Environments*, Norwell, MA: Kluwer Academic Publisher[in press]

Bergen, J.R. and Adelson, E.H. (1988), Visual texture segmentation and early vision. *Nature*, 333, p363-364

Malik, J. and Perona, P. (1990), Preattentive texture discrimination with early vision mechanisms. *J. Opt. Soc. Am.*, A Vol.7, No.5, p923-932

Cumming, B.G., Johnston, E.B., and Parker, A.J. (1993), Effects of different texture cues on curved surfaces viewed stereoscopically. *Vision Res.* Vol.33, No5, p827-838

Todd, J.T. and Akerstrom, R.A. (1987), Perception of three-dimensional form from patterns of optical texture. *Journal of Experimental Psychology*, vol.13, No.2, p242-255,

Turner, M.R., Gerstein, G.L., and Bajcsy, R. (1991), Underestimation of visual texture slant by human observers: a model. *Biol. Cybern.* 65, p215-226

Bülthoff, H.H. (1991), Shape from X: Psychophysics and computation. In Landy, M.S., et al.(ed.) *Computational Models of Visual Processing*, Cambridge, MA: MIT press, p305-330